# Synchronized Auditory and Cognitive 40 Hz Attentional Streams, and the Impact of Rhythmic Expectation on Auditory Scene Analysis

**Bill Baird**
Dept Mathematics, U.C.Berkeley, Berkeley, Ca. 94720.
baird@math.berkeley.edu

## Abstract

We have developed a neural network architecture that implements a theory of attention, learning, and trans-cortical communication based on adaptive synchronization of 5-15 Hz and 30-80 Hz oscillations between cortical areas. Here we present a specific higher order cortical model of attentional networks, rhythmic expectancy, and the interaction of higher-order and primary cortical levels of processing. It accounts for the "mismatch negativity" of the auditory ERP and the results of psychological experiments of Jones showing that auditory stream segregation depends on the rhythmic structure of inputs. The timing mechanisms of the model allow us to explain how relative timing information such as the relative order of events between streams is lost when streams are formed. The model suggests how the theories of auditory perception and attention of Jones and Bregman may be reconciled.

## 1 Introduction

Amplitude patterns of synchronized "gamma band" (30 to 80 Hz) oscillation have been observed in the ensemble activity (local field potentials) of vertebrate olfactory, visual, auditory, motor, and somatosensory cortex, and in the retina, thalamus, hippocampus, reticular formation, and EMG. Such activity has not only been found in primates, cats, rabbits and rats, but also insects, slugs, fish, amphibians, reptiles, and birds. This suggests that gamma oscillation may be as fundamental to neural processing at the network level as action potentials are at the cellular level.

We have shown how oscillatory associative memories may be coupled to recognize and generate sequential behavior, and how a set of novel mechanisms utilizing these complex dynamics can be configured to solve attentional and perceptual processing problems. For pointers to full treatment with mathematics and complete references see [Baird et al., 1994]. An important element of intra-cortical communication in the brain, and between modules in this architecture, is the ability of a module to detect and respond to the proper input signal from a particular module, when inputs from other modules which are irrelevant to the present computation are contributing crosstalk noise. We have demonstrated that selective control of synchronization, which we hypothesize to be a model of "attention", can be used to solve this coding problem and control program flow in an architecture with dynamic attractors [Baird et al., 1994].

Using dynamical systems theory, the architecture is constructed from recurrently interconnected oscillatory associative memory modules that model higher order sensory and motor areas of cortex. The modules learn connection weights between themselves which cause the system to evolve under a 5-20 Hz clocked sensory-motor processing cycle by a sequence

of transitions of synchronized 30-80 Hz oscillatory attractors within the modules. The architecture employs selective"attentional" control of the synchronization of the 30-80 Hz gamma band oscillations between modules to direct the flow of computation to recognize and generate sequences. The 30-80 Hz attractor amplitude patterns code the information content of a cortical area, whereas phase and frequency are used to "softwire" the network, since only the synchronized areas communicate by exchanging amplitude information. The system works like a broadcast network where the unavoidable crosstalk to all areas from previous learned connections is overcome by frequency coding to allow the moment to moment operation of attentional communication only between selected task-relevant areas.

The behavior of the time traces in different modules of the architecture models the temporary appearance and switching of the synchronization of 5-20 and 30-80 Hz oscillations between cortical areas that is observed during sensorimotor tasks in monkeys and numans. The architecture models the 5-20 Hz evoked potentials seen in the EEG as the control signals which determine the sensory-motor processing cycle. The 5-20 Hz clocks which drive these control signals in the architecture model thalamic pacemakers which are thought to control the excitability of neocortical tissue through similar nonspecific biasing currents that cause the cognitive and sensory evoked potentials of the EEG. The 5-20 Hz cycles "quantize time" and form the basis of derived somato-motor rhythms with periods up to seconds that entrain to each other in motor coordination and to external rhythms in speech perception [Jones et al., 1981].

## 1.1 Attentional Streams of Synchronized 40 Hz Activity

There is extensive evidence for the claim of the model that the 30-80 Hz gamma band activity in the brain accomplishes attentional processing, since 40 Hz appears in cortex when and where attention is required. For example, it is found in somatosensory, motor and premotor cortex of monkeys when they must pick a rasin out of a small box, but not when a habitual lever press delivers the reward. In human attention experiments, 30-80 Hz activity goes up in the contralateral auditory areas when subjects are instructed to pay attention to one ear and not the other. Gamma activity declines in the dominant hemisphere along with errors in a learnable target and distractors task, but not when the distractors and target vary at random on each trial. Anesthesiologists use the absence of 40 Hz activity as a reliable indicator of unconsciousness. Recent work has shown that cats with convergent and divergent strabismus who fail on tasks where perceptual binding is required also do not exhibit cortical synchrony. This is evidence that gamma synchronization is perceptually functional and not epiphenomenal.

The architecture illustrates the notion that synchronization of gamma band activity not only"binds" the features of inputs in primary sensory cortex into "objects", but further binds the activity of an attended object to oscillatory activity in associational and higher-order sensory and motor cortical areas to create an evolving attentional network of intercommunicating cortical areas that directs behavior. The binding of sequences of attractor transitions between modules of the architecture by synchronization of their activity models the physiological mechanism for the formation of perceptual and cognitive "streams" investigated by Bregman [Bregman, 1990], Jones [Jones et al., 1981], and others. In audition, according to Bregman's work, successive events of a sound source are bound together into a distinct sequence or "stream" and segregated from other sequences so that one pays attention to only one sound source at a time (the cocktail party problem). Higher order cortical or "cognitive" streams are in evidence when subjects are unable to recall the relative order of the telling of events between two stories told in alternating segments.

MEG tomographic observations show large scale rostral to caudal motor-sensory sweeps of coherent thalamo-cortical 40Hz activity accross the entire brain, the phase of which is reset by sensory input in waking, but not in dream states [Llinas and Ribary, 1993]. This suggests an inner higher order "attentional stream" is constantly cycling between motor (rostral) and sensory (caudal) areas in the absence of input. It may be interrupted by input "pop out" from primary areas or it may reach down as a "searchlight" to synchronize with particular ensembles of primary activity to be attended.

## 2   Jones Theory of Dynamic Attention

Jones [Jones et al., 1981] has developed a psychological theory of attention, perception, and motor timing based on the hypothesis that these processes are organized by neural rhythms in the range of 10 to .5 Hz – the range within which subjects perceive periodic events as a rhythm. These rhythms provide a multiscale representation of time and selectively synchronize with the prominant periodicities of an input to provide a temporal expectation mechanism for attention to target particular points in time.

For example, some work suggests that the accented parts of speech create a rhythm to which listeners entrain. Attention can then be focused on these expected locations as recognition anchor points for inference of less prominant parts of the speech stream. This is the temporal analog of the body centered spatial coordinate frame and multiscale covert attention window system in vision. Here the body centered temporal coordinates of the internal time base orient by entrainment to the external rhythm, and the window of covert temporal attention can then select a level of the multiscale temporal coordinates.

In this view, just as two cortical areas must synchronize to communicate, so must two nervous systems. Work using frame by frame film analysis of human verbal interaction, shows evidence of "interactional synchrony" of gesture and body movement changes and EEG of both speaker and listener with the onsets of phonemes in speech at the level of a 10 Hz "microrhythm" – the base clock rate of our models. Normal infants synchronize their spontaneous body flailings at this 10 Hz level to the mothers voice accents, while autistic and schitzophrenic children fail to show interactional synchrony. Autistics are unable to tap in time to a metronome.

Neural expectation rhythms that support Jones' theory have been found in the auditory EEG. In experiments where the arrival time of a target stimulus is regular enough to be learned by an experimental subject, it has been shown that the 10 Hz activity *in advance of the stimulus* becomes phase locked to that expected arrival time. This fits our model of rhythmic expectation where the 10 Hz rhythm is a fast base clock that is shifted in phase and frequency to produce a match in timing between the stimulus arrival and the output of longer period cycles derived from this base clock.

## 2.1 Mismatch Negativity

The "mismatch negativity" (MNN) [Naatanen, 1992] of the auditory evoked potential appears to be an important physiological indicator of the action of a neural expectancy system like that proposed by Jones. It has been localized to areas within primary auditory cortex by MEG studies [Naatanen, 1992] and it appears as an increased negativity of the ERP in the region of the N200 peak whenever a psychologically discriminable deviation of a repetitive auditory stimulus occurs. Mismatch is caused by deviations in onset or offset time, rise time, frequency, loudness, timbre, phonetic structure, or spatial location of a tone in the sequence. The mismatch is abolished by blockers of the action of NMDA channels [Naatanen, 1992] which are important for the synaptic changes underlying the kind of Hebbian learning which is used in the model.

MNN is not a direct function of echoic memory because it takes several repetitions for the expectancy to begin to develop, and it decays in 2 - 4 seconds. It appears only for repetition periods greater that 50-100 msec and less than 2-4 seconds. Thus the time scale of its operation is in the appropriate range for Jones' expectancy system. Stream formation also takes several cycles of stimulus repetition to build up over 2-4 seconds and decays away within 2-4 seconds in the absence of stimulation. Those auditory stimulus features which cause streaming are also features which cause mismatch. This supports the hypothesis in the model that these phenomena are functionally related.

Finally, MNN can occur independent of attention – while a subject is reading or doing a visual discrimination task. This implies that the auditory system at least must have its own timing system that can generate timing and expectancies independent of other behavior. We can talk or do internal verbal thinking while doing other tasks. A further component of this negativity appears in prefrontal cortex and is thought by Nataanen to initiate attentional switching toward the deviant event causing perceptual "pop out" [Naatanen, 1992].

Stream formation is known to affect rhythm perception. The galloping rhythm of high H and low L tones – HLH-HLH-HLH, for example becomes two separate isochronous rhythmic streams of H-H-H-H and L—L—L—L when the H and L tones are spread far enough apart [Bregman, 1990]. Evidence for the effect of input rhythms on stream formation, however, is more sparse, and we focus here on the simulation of a particular set of experiments by Jones [Jones et al., 1981] and Bregman [Bregman, 1990] where this effect has been demonstrated.

## 2.2 Jones-Bregman Experiment

Jones [Jones et al., 1981] replicated and altered a classic streaming experiment of Bregman and Rudnicky [Bregman, 1990], and found that their result depended on a specific choice of the rhythm of presentation. The experiment required human subjects to determine of the order of presentation of a pair of high target tones AB or BA of slightly different frequencies. Also presented before and after the target tones were a series of identical much lower frequency tones called the capture tones CCC and two identical tones of intermediate fre-

quency before and after the target tones called the flanking tones F - CCCFABFCCC. Bregman and Rudnicky found that target order determination performance was best when the capture tones were near to the flanking tones in frequency, and deteriorated as the captor tones were moved away. Their explanation was that the flanking tones were captured by the background capture tone stream when close in frequency, leaving the target tones to stand out by themselves in the attended stream. When the captor tones were absent or far away in frequency, the flanking tones were included in the attended stream and obscured the target tones.

Jones noted that the flanking tones and the capture stream were presented at a stimulus onset rate of one per 240 ms and the targets appeared at 80 ms intervals. In her experiments, when the captor and flanking tones were given a rhythm in common with the targets, no effect of the distance of captor and flanking tones appeared. This suggested that rhythmic distinction of targets and distractors was necessary in addition to the frequency distinction to allow selective attention to segregate out the target stream. Because performance in the single rhythm case was worse than that for the control condition without captors, it appeared that no stream segregation of targets and captors and flanking tones was occurring until the rhythmic difference was added. *From this evidence we make the assumption in the model that the distance of a stimulus in time from a rhythmic expectancy acts like the distance between stimuli in pitch, loudness, timbre, or spatial location as factor for the formation of separate streams.*

## 3   Architecture and Simulation

To implement Jones's theory in the model and account for her data, subsets of the oscillatory modules are dedicated to form a rhythmic temporal coordinate frame or time base by dividing down a thalamic 10 Hz base clock rate in steps from 10 to .5 Hz. Each derived clock is created by an associative memory module that has been specialized to act stereotypically as a counter or shift register by repeatedly cycling through all its attractors at the rate of one for each time step of its clock. Its overall cycle time is therefore determined by the number of attractors. Each cycle is guaranteed to be identical, as required for clocklike function, because of the strong attractors that correct the perturbing effect of noise. Only one step of the cycle can send output back to primary cortex - the one with the largest weight from receiving the most match to incoming stimuli. Each clock derived in this manner from a thalamic base clock will therefore phase reset itself to get the best match to incoming rhythms. The match can be further refined by frequency and phase entrainment of the base clock itself.

Three such counters are sufficient to model the rhythms in Jones' experiment as shown in the architecture of figure 1. The three counters divide the 12.5 Hz clock down to 6.25 and 4.16 Hz. The first contains one attractor at the base clock rate which has adapted to entrain to the 80 msec period of target stimulation (12.5 Hz). The second cycles at $12.5/2 = 6.25$ Hz, alternating between two attractors, and the third steps through three attractors, to cycle at $12.5/3 = 4.16$ Hz, which is the slow rhythm of the captor tones.

The modules of the time base send their internal 30-80 Hz activity to primary auditory cortex in 100msec bursts at these different rhythmic rates through fast adapting connections (which would use NMDA channels in the brain) that continually attempt to match incoming stimulus patterns using an incremental Hebbian learning rule. The weights decay to zero over 2-4 sec to simulate the data on the rise and fall of the mismatch negativity. These weights effectively compute a low frequency discrete Fourier transform over a sliding window of several seconds, and the basic periodic structure of rhythmic patterns is quickly matched. This serves to establish a quantized temporal grid of expectations against which expressive timing deviations in speech and music can be experienced.

Following Jones [Jones et al., 1981], we hypothesize that this happens automatically as a constant adaptation to environmental rhythms, as suggested by the mismatch negativity. Retained in these weights of the timebase is a special kind of short term memory of the activity which includes temporal information since the timebase will partially regenerate the previous activity in primary cortex *at the expected recurrence time*. This top-down input causes enchanced sensitivity in target units by increasing their gain. Those patterns which meet these established rhythmic expectancy signals in time are thereby boosted in amplitude and pulled into synchrony with the 30-80 Hz attentional searchlight stream to become part of the attentional network sending input to higher areas. In accordance with Jones' theory, voluntary top-down attention can probe input at different hierarchical levels of periodicity by selectively synchronizing a particular cortical column in the time base set to the 40 Hz frequency of the inner attention stream. Then the searchlight into primary cortex is synchro-

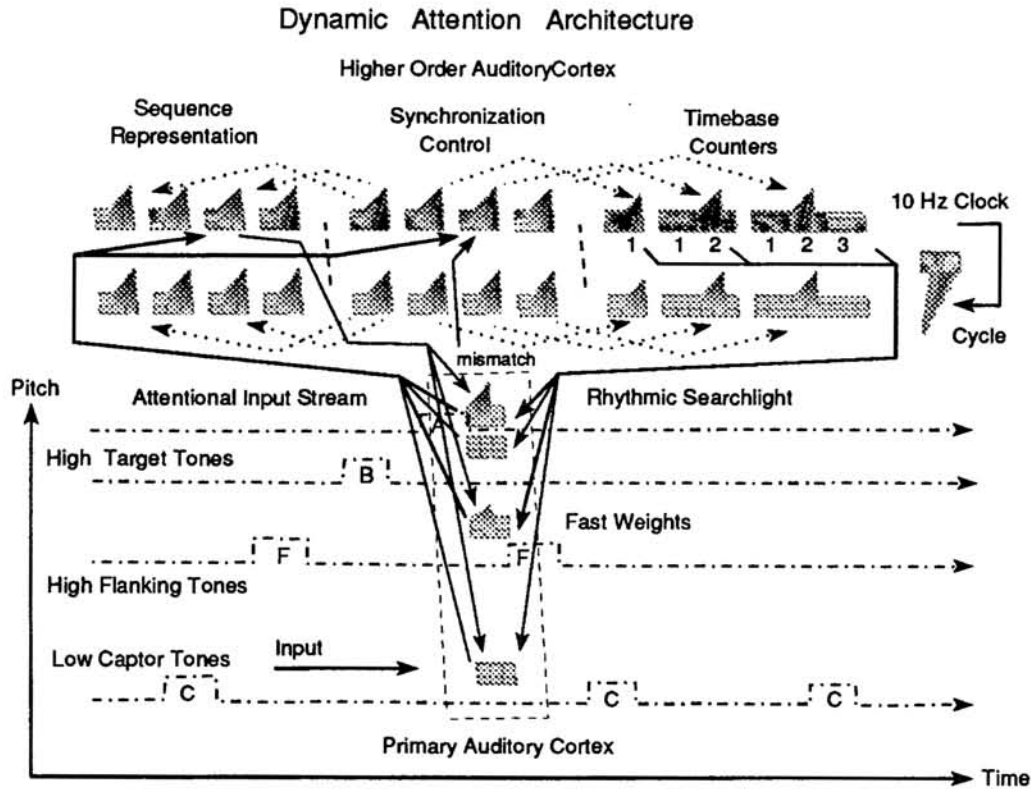

Figure 1: Horizontally arrayed units at the top model higher order auditory and motor cortical columns which are sequentially clocked by the (thalamic) base clock on the right to alternate attractor transitions between upper hidden (motor) and lower context (sensory) layers to act as an Elman net. Three cortical regions are shown – sequence representation memory, attentional synchronization control, and a rhythmic timebase of three counters. The hidden and context layers consist of binary "units" composed of two oscillatory attractors. Activity levels oscillate up and down through the plane of the paper. Dotted lines show frequency shifting outputs from the synchronization (attention) control modules. The lower vertical set of units is a sample of primary auditory cortex frequency channels at the values used in the Jones-Bregman experiment. The dashed lines show the rhythmic pattern of the target, flanking, and captor tones moving in time from left to right to impact on auditory cortex.

nizing and reading in activity occuring at the peaks of that particular time base rhythm.

## 3.1 Cochlear and Primary Cortex Model

At present, we have modeled only the minimal aspects of primary auditory cortex sufficient to qualitatively simulate the Jones-Bregman experiment, but the principles at work allow expansion to larger scale models with more stimulus features. We simulate four sites in auditory cortex corresponding to the four frequencies of stimuli used in the experiment, as shown in figure 1. There are two close high frequency target tones, one high flanking frequency location, and the low frequency location of the captor stream. These cortical locations are modeled as oscillators with the same equations used for associative memory modules [Baird et al., 1994], with full linear cross coupling weights. This lateral connectivity is sufficient to promote synchrony among simultaneously activated oscillators, but insufficient to activate them strongly in the absence of external input. This makes full synchrony of activated units the default condition in the model cortex, as in Brown's model [Brown and Cooke, 1996], so that the background activation is coherent, and can be read into higher order cortical levels which synchronize with it. The system assumes that all input is due to the same environmental source in the absence of evidence for segregation [Bregman, 1990].

Brown and Cooke [Brown and Cooke, 1996] model the cochlear and brainstem nuclear output as a set of overlapping bandpass ("gammatone") filters consistent with auditory nerve responses and psychophysical "critical bands". A tone can excite several filter outputs at once. We approximate this effect of the gammatone filters as a lateral fan out of input activations with weights that spread the activation in the same way as the overlapping gammatone

filters do.

Experiments show that the intrinsic resonant or "natural" frequencies or "eigenfrequencies" of cortical tissue within the 30-80 Hz gamma band vary within individuals on different trials of a task, and that neurotransmitters can quickly alter these resonant frequencies of neural clocks. Following the evidence that the oscillation frequency of binding in vision goes up with the speed of motion of an object, we assume that unattended activity in auditory cortex synchronizes at a default background frequency of 35 Hz, while the higher order attentional stream is at a higher frequency of 40 Hz. Just as fast motion in vision can cause stimulus driven capture of attention, we hypothesize that expectancy mismatch in audition causes the deviant activity to be boosted above the default background frequency to facilitate synchronization with the attentional stream at 40 Hz. This models the mechanism of involuntary stimulus driven attentional "pop out". Multiple streams of primary cortex activity synchronized at different eigenfrequencies can be selectively attended by uniformly sweeping the eigenfrequencies of all primary ensembles through the passband of the 40 Hz higher order attentional stream to "tune in" each in turn as a radio reciever does.

Following, but modifing the approach of Brown and Cooke [Brown and Cooke, 1996], the core of our primary cortex stream forming model is a fast learning rule that reduces the lateral coupling and (in our model) spreads apart the intrinsic cortical frequencies of sound frequency channels that do not exhibit the same amplitude of activity at the same time. This coupling and eigenfrequency difference recovers between onsets. In the absence of lateral synchronizing connections or coherent top down driving, synchrony between cortical streams is rapidly lost because of their distant resonant frequencies. Activity not satisfying the Gestalt principle of "common fate" [Bregman, 1990] is thus decorrelated.

The trade off of the effect of temporal and sound frequency proximity on stream segregation follows because close stimulus frequencies excite each other's channel filters. Each produces a similar output in the other, and their activitites are not decorrelated by coupling reduction and resonant frequency shifts. On the other hand, to the extent that they are distant enough in sound frequency, each tone onset weakens the weights and shifts the eigenfrequencies of the other channels that are not simultaneously active. This effect is greater, the faster the presentation rate, because the weight recovery rate is overcome. This recovery rate can then be adjusted to yield stream segregation at the rates reported by van Noorden [Bregman, 1990] for given sound frequency separations.

## 3.2  Sequential Grouping by Coupling and Resonant Frequency Labels

In the absence of rhythmic structure in the input, the temporary weights and resonant frequency "labels" serve as a short term "stream memory" to bridge time (up to 4 seconds) so that the next nearby input is "captured" or "sequentially bound" into the same ensemble of synchronized activity. This pattern of synchrony in primary cortex has been made into a temporary attractor by the temporary weight and eigenfrequency changes from the previous stimulation. This explains the single tone capture experiments where a series of identical tones captures later nearby tones. For two points in time to be sequentially grouped by this mechanism, there is no need for activity to continue between onsets as in Browns model [Brown and Cooke, 1996], or to be held in multiple spatial locations as Wang [Wang, 1995] does. Since the gamma band response to a single auditory input onset lasts only 100 - 150 ms, there is no 40 Hz activity available in primary cortex (at most stimulus rates) for succesive inputs to synchronize with for sequential binding by these mechanisms.

Furthermore, the decorrelation rule, when added to the mechanism of timing expectancies, explains the loss of relative timing (order) between streams, since the lateral connections that normally broadcast actual and expected onsets accross auditory cortex, are cut between two streams by the decorrelating weight reduction. Expected and actual onset events in different streams can no longer be directly (locally) compared. Experimental evidence for the broadcast of expectancies comes from the fast generalization to other frequencies of a learned expectancy for the onset time of a tone of a particular frequency (Schreiner lab - personal communication).

When rhythmic structure is present, the expectancy system becomes engaged, and this becomes an additional feature dimension along which stimuli can be segregated. Distance from expected *timing* as well as sound quality is now an added factor causing stream formation by decoupling and eigenfrequency shift. Feedback of expected input can also partially "fill in" missing input for a cycle or two so that the expectancy protects the binding of features of a stimulus and stabilizes a perceptual stream across seconds of time.

### 3.3 Simulation of the Jones-Bregman Experiment

Figure 2 shows the architecture used to simulate the Jones-Bregman experiment. The case shown is where the flanking tones are in the same stream as the targets because the captor stream is at the lower sound frequency channel. At the particular point in time shown here, the first flanking tone has just finished, and the first target tone has arrived. Both channels are therfore active, and synchronized with the attentional stream into the higher order sequence recognizer.

Our mechanistic explanation of the Bregman result is that the early standard target tones arriving at the 80 msec rate first **prime** the dynamic attention system by setting the 80 msec clock to oscillate at 40 Hz and depressing the oscillation frequency of other auditory cortex background activity. Then the slow captor tones at the 240 msec period establish a background stream at 30 Hz with a rhythmic expectancy that is later violated by the appearance of the fast target tones. These now fall outside the correlation attractor basin of the background stream because the mismatch increases their cortical oscillation frequency. They are explicitly brought into the 40 Hz foreground frequency by the mismatch pop out mechanism. This allows the attentional stream into the Elman sequence recognition units to synchronize and read in activity due to the target tones for order determination. It is assisted by the timebase searchlight at the 80 msec period which synchronizes and enhances activity arriving at that rhythm. In the absence of a rhythmic distinction for the target tones, their sound frequency difference alone is insufficient to separate them from the background stream, and the targets cannot be reliably discriminated.

In this simulation, the connections to the first two Elman associative memory units are hand wired to the A and B primary cortex oscillators to act as a latching, order determining switch. If synchronized to the memory unit at the attentional stream frequency, the A target tone oscillator will drive the first memory unit into the 1 attractor which then inhibits the second unit from being driven to 1 by the B target tone. The second unit has similar wiring from the B tone oscillator, so that the particular higher order (intermediate term) memory unit which is left in the 1 state after a trial indicates to the rest of the brain which tone came first. The flanking and high captor tone oscillator is connected equally to both memory units, so that a random attractor transition occurs before the targets arrive, when it is interfering at the 40 Hz attentional frequency, and poor order determination results. If the flanking tone oscillator is in a separate stream along with the captor tones at the background eigenfrequency of 35 Hz, it is outside the recieving passband of the memory units and cannot cause a spurious attractor transition.

This architecture demonstrates mechanisms that integrate the theories of Jones and Bregman about auditory perception. Stream formation is a preattentive process that works well on non-rhythmic inputs as Bregman asserts, but an equally primary and preattentive rhythmic expectancy process is also at work as Jones asserts and the mismatch negativity indicates. This becomes a factor in stream formation when rhythmic structure is present in stimuli as demonstrated by Jones.

## References

[Baird et al., 1994] Baird, B., Troyer, T., and Eeckman, F. H. (1994). Gramatical inference by attentional control of synchronization in an oscillating elman network. In Hanson, S., Cowan, J., and Giles, C., editors, *Advances in Neural Information Processing Systems 6*, pages 67–75. Morgan Kaufman.

[Bregman, 1990] Bregman, A. S. (1990). *Auditory Scene Analysis*. MIT Press, Cambridge.

[Brown and Cooke, 1996] Brown, G. and Cooke, M. (1996). A neural oscillator model of auditory stream segregation. In *IJCAI Workshop on Computational Auditory Scene Analysis*. to appear.

[Jones et al., 1981] Jones, M., Kidd, G., and Wetzel, R. (1981). Evidence for rhythmic attention. *Journal of Experimental Psychology: Human Perception and Performance*, 7:1059–1073.

[Llinas and Ribary, 1993] Llinas, R. and Ribary, U. (1993). Coherent 40-hz oscillation characterizes dream state in humans. *Proc. Natl. Acad. Sci. USA*, 90:2078–2081.

[Naatanen, 1992] Naatanen, R. (1992). *Attention and Brain Function*. Erlbaum, New Jersey.

[Wang, 1995] Wang, D. (1995). An oscillatory correlation theory of temporal pattern segmentation. In Covey, E., Hawkins, H., McMullen, T., and Port, R., editors, *Neural Representations of Temporal Patterns*. Plenum. to appear.